# Optimal ROC Curve for a Combination of Classifiers

**Marco Barreno**　　　**Alvaro A. Cárdenas**　　　**J. D. Tygar**

Computer Science Division
University of California at Berkeley
Berkeley, California 94720
{barreno,cardenas,tygar}@cs.berkeley.edu

## Abstract

We present a new analysis for the combination of binary classifiers. Our analysis makes use of the Neyman-Pearson lemma as a theoretical basis to analyze combinations of classifiers. We give a method for finding the optimal decision rule for a combination of classifiers and prove that it has the optimal ROC curve. We show how our method generalizes and improves previous work on combining classifiers and generating ROC curves.

## 1   Introduction

We present an optimal way to combine binary classifiers in the Neyman-Pearson sense: for a given upper bound on false alarms (false positives), we find the set of combination rules maximizing the detection rate (true positives). This forms the optimal ROC curve of a combination of classifiers.

This paper makes the following original contributions: (1) We present a new method for finding the meta-classifier with the optimal ROC curve. (2) We show how our framework can be used to *interpret, generalize,* and *improve* previous work by Provost and Fawcett [1] and Flach and Wu [2]. (3) We present experimental results that show our method is practical and performs well, even when we must estimate the distributions with insufficient data.

In addition, we prove the following results: (1) We show that the optimal ROC curve is composed in general of $2^n + 1$ different decision rules and of the interpolation between these rules (over the space of $2^{2^n}$ possible Boolean rules). (2) We prove that our method is optimal in this space. (3) We prove that the Boolean AND and OR rules are always part of the optimal set for the special case of independent classifiers (though in general we make no independence assumptions). (4) We prove a sufficient condition for Provost and Fawcett's method to be optimal.

## 2   Background

Consider classification problems where examples from a space of inputs $\mathcal{X}$ are associated with binary labels $\{0, 1\}$ and there is a fixed but unknown probability distribution $\mathbf{P}(x, c)$ over examples $(x, c) \in \mathcal{X} \times \{0, 1\}$. $H_0$ and $H_1$ denote the events that $c = 0$ and $c = 1$, respectively.

A binary classifier is a function $f : \mathcal{X} \rightarrow \{0, 1\}$ that predicts labels on new inputs. When we use the term "classifier" in this paper we mean binary classifier. We address the problem of combining results from $n$ *base classifiers* $f_1, f_2, \ldots, f_n$. Let $Y_i = f_i(X)$ be a random variable indicating the output of classifier $f_i$ and $\mathbf{Y} \in \{0, 1\}^n = (Y_1, Y_2, \ldots, Y_n)$. We can characterize the performance of classifier $f_i$ by its *detection rate* (also *true positives*, or *power*) $P_{Di} = \Pr[Y_i = 1|H_1]$ and its *false alarm rate* (also *false positives*, or *test size*) $P_{Fi} = \Pr[Y_i = 1|H_0]$. In this paper we are concerned with *proper* classifiers, that is, classifiers where $P_{Di} > P_{Fi}$. We sometimes omit the subscript $i$.

The *Receiver Operating Characteristic (ROC) curve* plots $P_F$ on the $x$-axis and $P_D$ on the $y$-axis (*ROC space*). The point $(0,0)$ represents always classifying as 0, the point $(1,1)$ represents always classifying as 1, and the point $(0,1)$ represents perfect classification. If one classifier's curve has no points below another, it *weakly dominates* the latter. If no points are below and at least one point is strictly above, it *dominates* it. The line $y = x$ describes a classifier that is no better than chance, and every proper classifier dominates this line. When an ROC curve consists of a single point, we connect it with straight lines to $(0,0)$ and $(1,1)$ in order to compare it with others (see Lemma 1).

In this paper, we focus on base classifiers that occupy a single point in ROC space. Many classifiers have tunable parameters and can produce a continuous ROC curve; our analysis can apply to these cases by choosing representative points and treating each one as a separate classifier.

## 2.1 The ROC convex hull

Provost and Fawcett [1] give a seminal result on the use of ROC curves for combining classifiers. They suggest taking the convex hull of all points of the ROC curves of the classifiers. This *ROC convex hull (ROCCH)* combination rule interpolates between base classifiers $f_1, f_2, \ldots, f_n$, selecting (1) a single best classifier or (2) a randomization between the decisions of two classifiers for every false alarm rate [1]. This approach, however, is not optimal: as pointed out in later work by Fawcett, the Boolean *AND* and *OR* rules over classifiers can perform better than the ROCCH [3].

*AND* and *OR* are only 2 of $2^{2^n}$ possible Boolean rules over the outputs of $n$ base classifiers ($n$ classifiers $\Rightarrow 2^n$ possible outcomes $\Rightarrow 2^{2^n}$ rules over outcomes). We address finding optimal rules.

## 2.2 The Neyman-Pearson lemma

In this section we introduce Neyman-Pearson theory from the framework of statistical hypothesis testing [4, 5], which forms the basis of our analysis.

We test a null hypothesis $H_0$ against an alternative $H_1$. Let the random variable $\mathbf{Y}$ have probability distributions $P(\mathbf{Y}|H_0)$ under $H_0$ and $P(\mathbf{Y}|H_1)$ under $H_1$, and define the *likelihood ratio* $\ell(\mathbf{Y}) = P(\mathbf{Y}|H_1)/P(\mathbf{Y}|H_0)$. The Neyman-Pearson lemma states that the likelihood ratio test

$$\mathcal{D}(\mathbf{Y}) = \left\{ \begin{array}{ll} 1 & \text{if } \ell(\mathbf{Y}) > \tau \\ \gamma & \text{if } \ell(\mathbf{Y}) = \tau \\ 0 & \text{if } \ell(\mathbf{Y}) < \tau \end{array} \right. , \tag{1}$$

for some $\tau \in (0, \infty)$ and $\gamma \in [0, 1]$, is a most powerful test for its size: no other test has higher $P_D = \Pr[\mathcal{D}(\mathbf{Y}) = 1|H_1]$ for the same bound on $P_F = \Pr[\mathcal{D}(\mathbf{Y}) = 1|H_0]$. (When $\ell(\mathbf{Y}) = \tau$, $\mathcal{D} = 1$ with probability $\gamma$ and 0 otherwise.) Given a test size $\alpha$, we maximize $P_D$ subject to $P_F \leq \alpha$ by choosing $\tau$ and $\gamma$ as follows. First we find the smallest value $\tau^*$ such that $\Pr[\ell(\mathbf{Y}) > \tau^*|H_0] \leq \alpha$. To maximize $P_D$, which is monotonically nondecreasing with $P_F$, we choose the highest value $\gamma^*$ that satisfies $\Pr[\mathcal{D}(\mathbf{Y}) = 1|H_0] = \Pr[\ell(\mathbf{Y}) > \tau^*|H_0] + \gamma^* \Pr[\ell(\mathbf{Y}) = \tau^*|H_0] \leq \alpha$, finding $\gamma^* = (\alpha - \Pr[\ell(\mathbf{Y}) > \tau^*|H_0])/\Pr[\ell(\mathbf{Y}) = \tau^*|H_0]$.

# 3 The optimal ROC curve for a combination of classifiers

We characterize the optimal ROC curve for a decision based on a combination of arbitrary classifiers—for any given bound $\alpha$ on $P_F$, we maximize $P_D$. We frame this problem as a Neyman-Pearson hypothesis test parameterized by the choice of $\alpha$. We assume nothing about the classifiers except that each produces an output in $\{0, 1\}$. In particular, we do not assume the classifiers are independent or related in any way.

Before introducing our method we analyze the one-classifier case ($n = 1$).

**Lemma 1** *Let $f_1$ be a classifier with performance probabilities $P_{D1}$ and $P_{F1}$. Its optimal ROC curve is a piecewise linear function parameterized by a free parameter $\alpha$ bounding $P_F$: for $\alpha < P_{F1}$, $P_D(\alpha) = (P_{D1}/P_{F1})\alpha$, and for $\alpha > P_{F1}$, $P_D(\alpha) = [(1-P_{D1})/(1-P_{F1})](\alpha - P_{F1}) + P_{D1}$.*

**Proof.** When $\alpha < P_{F1}$, we can obtain a likelihood ratio test by setting $\tau^* = \ell(1)$ and $\gamma^* = \alpha/P_{F1}$, and for $\alpha > P_{F1}$, we set $\tau^* = \ell(0)$ and $\gamma^* = (\alpha - P_{F1})/(1 - P_{F1})$. □

The intuitive interpretation of this result is that to decrease or increase the false alarm rate of the classifier, we randomize between using its predictions and always choosing 1 or 0. In ROC space, this forms lines interpolating between $(P_{F1}, P_{D1})$ and $(1, 1)$ or $(0, 0)$, respectively.

To generalize this result for the combination of $n$ classifiers, we require the distributions $P(\mathbf{Y}|H_0)$ and $P(\mathbf{Y}|H_1)$. With this information we then compute and sort the likelihood ratios $\ell(\mathbf{y})$ for all *outcomes* $\mathbf{y} \in \{0, 1\}^n$. Let $\mathcal{L}$ be the list of likelihood ratios ranked from low to high.

**Lemma 2** *Given any $0 \leq \alpha \leq 1$, the ordering $\mathcal{L}$ determines parameters $\tau^*$ and $\gamma^*$ for a likelihood ratio test of size $\alpha$.*

Lemma 2 sets up a classification rule for each interval between likelihoods in $\mathcal{L}$ and interpolates between them to create a test with size exactly $\alpha$. Our meta-classifier does this for any given bound on its false positive rate, then makes predictions according to Equation 1. To find the ROC curve for our meta-classifier, we plot $P_D$ against $P_F$ for all $0 \leq \alpha \leq 1$. In particular, for each $\mathbf{y} \in \{0, 1\}^n$ we can compute $\Pr[\ell(\mathbf{Y}) > \ell(\mathbf{y})|H_0]$, which gives us one value for $\tau^*$ and a point in ROC space ($P_F$ and $P_D$ follow directly from $\mathcal{L}$ and $P$). Each $\tau^*$ will turn out to be the slope of a line segment between adjacent vertices, and varying $\gamma^*$ interpolates between the vertices. We call the ROC curve obtained in this way the *LR-ROC*.

**Theorem 1** *The LR-ROC weakly dominates the ROC curve of any possible combination of Boolean functions $g : \{0, 1\}^n \rightarrow \{0, 1\}$ over the outputs of $n$ classifiers.*

**Proof.**  Let $\alpha'$ be the probability of false alarm $P_F$ for $g$. Let $\tau^*$ and $\gamma^*$ be chosen for a test of size $\alpha'$. Then our meta-classifier's decision rule is a likelihood ratio test. By the Neyman-Pearson lemma, no other test has higher power for any given size. Since ROC space plots power on the $y$-axis and size on the $x$-axis, this means that the $P_D$ for $g$ at $P_F = \alpha'$ cannot be higher than that of the LR-ROC. Since this is true at any $\alpha'$, the LR-ROC weakly dominates the ROC curve for $g$.  $\square$

## 3.1  Practical considerations

To compute all likelihood ratios for the classifier outcomes we need to know the probability distributions $P(\mathbf{Y}|H_0)$ and $P(\mathbf{Y}|H_1)$. In practice these distributions need to be estimated. The simplest method is to run the base classifiers on a training set and count occurrences of each outcome. It is likely that some outcomes will not occur in the training, or will occur only a small number of times. Our initial approach to deal with small or zero counts when estimating was to use add-one smoothing. In our experiments, however, simple special-case treatment of zero counts always produced better results than smoothing, both on the training set and on the test set. See Section 5 for details.

Furthermore, the optimal ROC curve may have a different likelihood ratio for each possible outcome from the $n$ classifiers, and therefore a different point in ROC space, so optimal ROC curves in general have up to $2^n$ points. This implies an exponential (in the number of classifiers) lower bound on the running time of any algorithm to compute the optimal ROC curve for a combination of classifiers. For a handful of classifiers, such a bound is not problematic, but it is impractical to compute the optimal ROC curve for dozens or hundreds of classifiers. (However, by computing and sorting the likelihood ratios we avoid a $2^{2^n}$-time search over all possible classification functions.)

## 4  Analysis

### 4.1  The independent case

In this section we take an in-depth look at the case of two binary classifiers $f_1$ and $f_2$ that are conditionally independent given the input's class, so that $P(Y_1, Y_2|H_c) = P(Y_1|H_c)P(Y_2|H_c)$ for $c \in \{0, 1\}$ (this section is the only part of the paper in which we make any independence assumptions). Since $Y_1$ and $Y_2$ are conditionally independent, we do not need the full joint distribution; we need only the probabilities $P_{D1}$, $P_{F1}$, $P_{D2}$, and $P_{F2}$ to find the combined $P_D$ and $P_F$. For example, $\ell(01) = ((1 - P_{D1})P_{D2})/((1 - P_{F1})P_{F2})$.

The assumption that $f_1$ and $f_2$ are conditionally independent and proper defines a partial ordering on the likelihood ratio: $\ell(00) < \ell(10) < \ell(11)$ and $\ell(00) < \ell(01) < \ell(11)$. Without loss of

Table 1: Two probability distributions.

| | Class 1 ($H_1$) $Y_1$ | | | Class 0 ($H_0$) $Y_1$ | | | Class 1 ($H_1$) $Y_1$ | | | Class 0 ($H_0$) $Y_1$ | |
|-------|-----|-------|-------|-----|-----|-------|-----|-----|-------|-----|-----|
| $Y_2$ | 0   | 1     | $Y_2$ | 0   | 1   | $Y_2$ | 0   | 1   | $Y_2$ | 0   | 1   |
| 0     | 0.2 | 0.375 | 0     | 0.5 | 0.1 | 0     | 0.2 | 0.1 | 0     | 0.1 | 0.3 |
| 1     | 0.1 | 0.325 | 1     | 0.3 | 0.1 | 1     | 0.2 | 0.5 | 1     | 0.5 | 0.1 |
| | (a) | | | | | | (b) | | | | |

generality, we assume $\ell(00) < \ell(01) < \ell(10) < \ell(11)$. This ordering breaks the likelihood ratio's range $(0, \infty)$ into five regions; choosing $\tau$ in each region defines a different decision rule.

The trivial cases $0 \leq \tau < \ell(00)$ and $\ell(11) < \tau < \infty$ correspond to always classifying as 1 and 0, respectively. $P_D$ and $P_F$ are therefore both equal to 1 and both equal to 0, respectively. For the case $\ell(00) \leq \tau < \ell(01)$, $\Pr[\ell(\mathbf{Y}) > \tau] = \Pr[\mathbf{Y} = 01 \vee \mathbf{Y} = 10 \vee \mathbf{Y} = 11] = \Pr[Y_1 = 1 \vee Y_2 = 1]$. Thresholds in this range define an OR rule for the classifiers, with $P_D = P_{D1} + P_{D2} - P_{D1}P_{D2}$ and $P_F = P_{F1} + P_{F2} - P_{F1}P_{F2}$. For the case $\ell(01) \leq \tau < \ell(10)$, we have $\Pr[\ell(\mathbf{Y}) > \tau] = \Pr[\mathbf{Y} = 10 \vee \mathbf{Y} = 11] = \Pr[Y_1 = 1]$. Therefore the performance probabilities are simply $P_D = P_{D1}$ and $P_F = P_{F1}$. Finally, the case $\ell(10) \leq \tau < \ell(11)$ implies that $\Pr[\ell(\mathbf{Y}) > \tau] = \Pr[\mathbf{Y} = 11]$, and therefore thresholds in this range define an AND rule, with $P_D = P_{D1}P_{D2}$ and $P_F = P_{F1}P_{F2}$. Figure 1a illustrates this analysis with an example.

The assumption of conditional independence is a sufficient condition for ensuring that the AND and OR rules improve on the ROCCH for $n$ classifiers, as the following result shows.

**Theorem 2** *If the distributions of the outputs of $n$ proper binary classifiers $Y_1, Y_2, \ldots, Y_n$ are conditionally independent given the instance class, then the points in ROC space for the rules AND $(Y_1 \wedge Y_2 \wedge \cdots \wedge Y_n)$ and OR $(Y_1 \vee Y_2 \vee \cdots \vee Y_n)$ are strictly above the convex hull of the ROC curves of the base classifiers $f_1, \ldots, f_n$. Furthermore, these Boolean rules belong to the LR-ROC.*

**Proof.** The likelihood ratio of the case when AND outputs 1 is given by $\ell(11 \cdots 1) = (P_{D1}P_{D2} \cdots P_{Dn})/(P_{F1}P_{F2} \cdots P_{Fn})$. The likelihood ratio of the case when OR does not output 1 is given by $\ell(00 \cdots 0) = [(1 - P_{D1})(1 - P_{D2}) \cdots (1 - P_{Dn})]/[(1 - P_{F1})(1 - P_{F2}) \cdots (1 - P_{Fn})]$. Now recall that for proper classifiers $f_i$, $P_{Di} > P_{Fi}$ and thus $(1 - P_{Di})/(1 - P_{Fi}) < 1 < P_{Di}/P_{Fi}$. It is now clear that $\ell(00 \cdots 0)$ is the smallest likelihood ratio and $\ell(11 \cdots 1)$ is the largest likelihood ratio, since others are obtained only by swapping $P_{(F,D)i}$ and $(1 - P_{(F,D)i})$, and therefore the OR and AND rules will always be part of the optimal set of decisions for conditionally independent classifiers. These rules are *strictly* above the ROCCH: because $\ell(11 \cdots 1) > P_{D1}/P_{D2}$, and $P_{D1}/P_{D2}$ is the slope of the line from $(0,0)$ to the first point in the ROCCH ($f_1$), the AND point must be above the ROCCH. A similar argument holds for OR since $\ell(00 \cdots 0) < (1 - P_{Dn})/(1 - P_{Fn})$. □

### 4.2 Two examples

We return now to the general case with no independence assumptions. We present two example distributions for the two-classifier case that demonstrate interesting results.

The first distribution appears in Table 1a. The likelihood ratio values are $\ell(00) = 0.4$, $\ell(10) = 3.75$, $\ell(01) = 1/3$, and $\ell(11) = 3.25$, giving us $\ell(01) < \ell(00) < \ell(11) < \ell(10)$. The three non-trivial rules correspond to the Boolean functions $Y_1 \vee \neg Y_2$, $Y_1$, and $Y_1 \wedge \neg Y_2$. Note that $Y_2$ appears only negatively despite being a proper classifier, and both the AND and OR rules are sub-optimal.

The distribution for the second example appears in Table 1b. The likelihood ratios of the outcomes are $\ell(00) = 2.0$, $\ell(10) = 1/3$, $\ell(01) = 0.4$, and $\ell(11) = 5$, so $\ell(10) < \ell(01) < \ell(00) < \ell(11)$ and the three points defining the optimal ROC curve are $\neg Y_1 \vee Y_2$, $\neg(Y_1 \oplus Y_2)$, and $Y_1 \wedge Y_2$ (see Figure 1b). In this case, an XOR rule emerges from the likelihood ratio analysis.

These examples show that for true optimal results it is not sufficient to use weighted voting rules $w_1Y_1 + w_2Y_2 + \cdots + w_nY_n \geq \tau$, where $w \in (0, \infty)$ (like some ensemble methods). Weighted voting always has AND and OR rules in its ROC curve, so it cannot always express optimal rules.

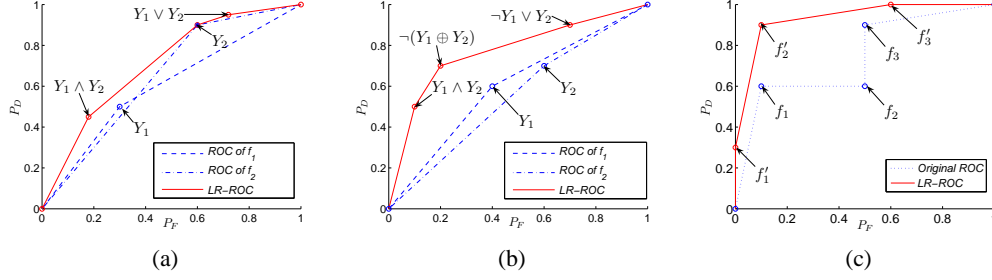

Figure 1: (a) ROC for two conditionally independent classifiers. (b) ROC curve for the distributions in Table 1b. (c) Original ROC curve and optimal ROC curve for example in Section 4.4.

## 4.3 Optimality of the ROCCH

We have seen that in some cases, rules exist with points strictly above the ROCCH. As the following result shows, however, there are conditions under which the ROCCH is optimal.

**Theorem 3** *Consider $n$ classifiers $f_1, \ldots, f_n$. The convex hull of points $(P_{Fi}, P_{Di})$ with $(0,0)$ and $(1,1)$ (the ROCCH) is an optimal ROC curve for the combination if $(Y_i = 1) \Rightarrow (Y_j = 1)$ for $i < j$ and the following ordering holds: $\ell(00 \cdots 0) < \ell(00 \cdots 01) < \ell(00 \cdots 011) < \cdots < \ell(1 \cdots 1)$.*

**Proof.** The condition $(Y_i = 1) \Rightarrow (Y_j = 1)$ for $i < j$ implies that we only need to consider $n + 2$ points in the ROC space (the two extra points are $(0,0)$ and $(1,1)$) rather than $2^n$. It also implies the following conditions on the joint distribution: $\Pr[Y_1 = 0 \wedge \cdots \wedge Y_i = 0 \wedge Y_{i+1} = 1 \wedge \cdots \wedge Y_n = 1|H_0] = P_{Fi+1} - P_{Fi}$, and $\Pr[Y_1 = 1 \wedge \cdots \wedge Y_n = 1|H_0] = P_{F1}$. With these conditions and the ordering condition on the likelihood ratios, we have $\Pr[\ell(\mathbf{Y}) > \ell(1 \cdots 1)|H_0] = 0$, and $\Pr[\ell(\mathbf{Y}) > \ell(\underbrace{0 \cdots 0}_{i} 1 \cdots 1)|H_0] = P_{Fi}$. Therefore, finding the optimal threshold of the likelihood ratio test for $P_{Fi-1} \leq \alpha < P_{Fi}$, we get $\tau^* = \ell(\underbrace{0 \cdots 0}_{i-1} 1 \cdots 1)$, and for $P_{Fi} \leq \alpha < P_{Fi+1}$, $\tau^* = \ell(\underbrace{0 \cdots 0}_{i} 1 \cdots 1)$. This change in $\tau^*$ implies that the point $P_{Fi}$ is part of the LR-ROC. Setting $\alpha = P_{Fi}$ (thus $\tau^* = \ell(\underbrace{0 \cdots 0}_{i} 1 \cdots 1)$ and $\gamma^* = 0$) implies $\Pr[\ell(\mathbf{Y}) > \tau^*|H_1] = P_{Di}$. $\qquad \square$

The condition $Y_i = 1 \Rightarrow Y_j = 1$ for $i < j$ is the same inclusion condition Flach and Wu use for repairing an ROC curve [2]. It intuitively represents the performance in ROC space of a single classifier with different operating points. The next section explores this relationship further.

## 4.4 Repairing an ROC curve

Flach and Wu give a voting technique to repair concavities in an ROC curve that generates operating points above the ROCCH [2]. Their intuition is that points underneath the convex hull can be mirrored to appear above the convex hull in much the same way as an improper classifier can be negated to obtain a proper classifier. Although their algorithm produces better ROC curves, their solution will often yield curves with new concavities (see for example Flach and Wu's Figure 4 [2]). Their algorithm has a similar purpose to ours, but theirs is a local greedy optimization technique, while our method performs a global search in order to find the best ROC curve.

Figure 1c shows an example comparing their method to ours. Consider the following probability distribution on a random variable $\mathbf{Y} \in \{0,1\}^2$: $P((00, 10, 01, 11)|H_1) = (0.1, 0.3, 0.0, 0.6)$, $P((00, 10, 01, 11)|H_0) = (0.5, 0.001, 0.4, 0.099)$. Flach and Wu's method assumes the original ROC curve to be repaired has three *models*, or operating points: $f_1$ predicts 1 when $\mathbf{Y} \in \{11\}$, $f_2$ predicts 1 when $\mathbf{Y} \in \{11, 01\}$, and $f_3$ predicts 1 when $\mathbf{Y} \in \{11, 01, 10\}$. If we apply Flach and Wu's repair algorithm, the point $f_2$ is corrected to the point $f_2'$; however, the operating points of $f_1$ and $f_3$ remain the same.

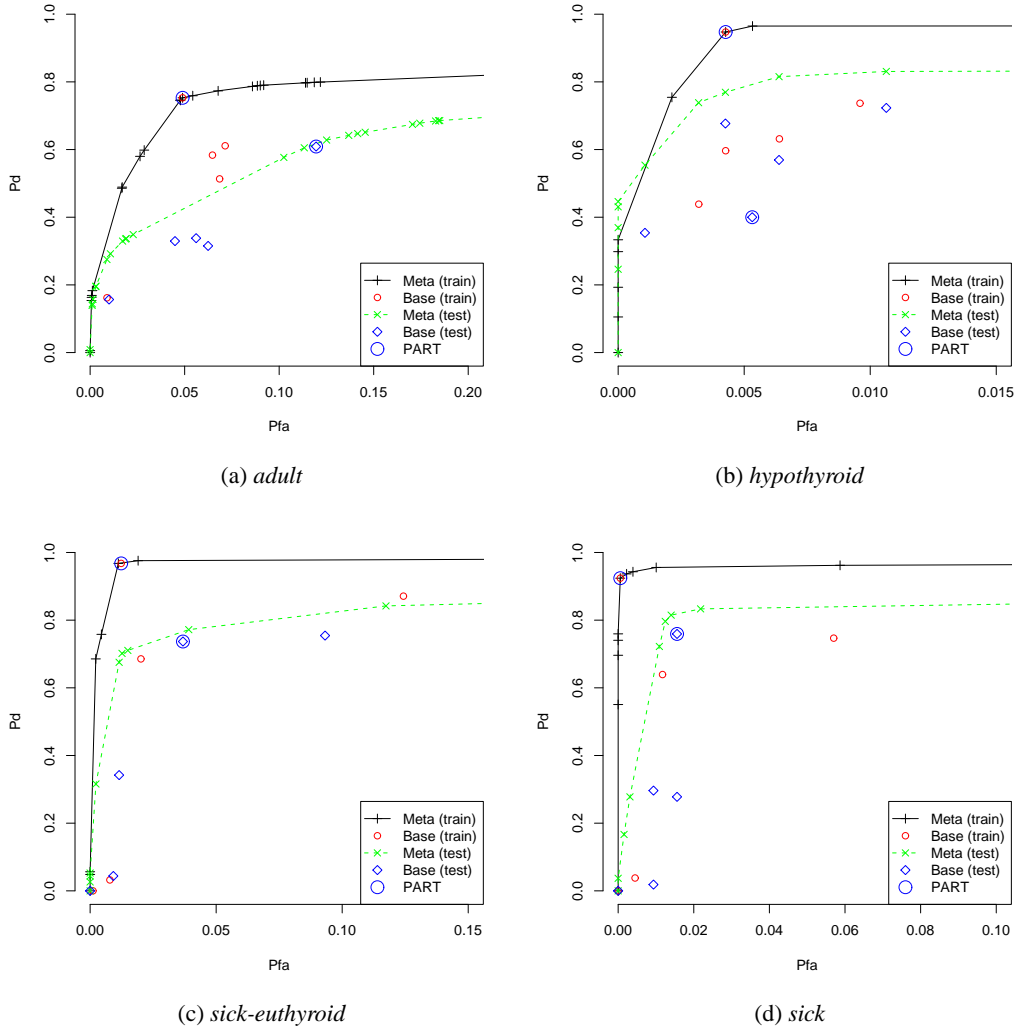

(a) *adult*

(b) *hypothyroid*

(c) *sick-euthyroid*

(d) *sick*

Figure 2: Empirical ROC curves for experimental results on four UCI datasets.

Our method improves on this result by ordering the likelihood ratios $\ell(01) < \ell(00) < \ell(11) < \ell(10)$ and using that ordering to make three different rules: $f_1'$ predicts 1 when $\mathbf{Y} \in \{10\}$, $f_2'$ predicts 1 when $\mathbf{Y} \in \{10, 11\}$, and $f_3'$ predicts 1 when $\mathbf{Y} \in \{10, 11, 00\}$.

## 5 Experiments

We ran experiments to test the performance of our combining method on the *adult*, *hypothyroid*, *sick-euthyroid*, and *sick* datasets from the UCI machine learning repository [6]. We chose five base classifiers from the YALE machine learning platform [7]: PART (a decision list algorithm), SMO (Sequential Minimal Optimization), SimpleLogistic, VotedPerceptron, and Y-NaiveBayes. We used default settings for all classifiers. The *adult* dataset has around 30,000 training points and 15,000 test points and the *sick* dataset has around 2000 training points and 700 test points. The others each have around 2000 points that we split randomly into 1000 training and 1000 test.

For each experiment, we estimate the joint distribution by training the base classifiers on a training set and counting the outcomes. We compute likelihood ratios for all outcomes and order them. When outcomes have no examples, we treat $\cdot/0$ as near-infinite and $0/\cdot$ as near-zero and define $0/0 = 1$.

We derive a sequence of decision rules from the likelihood ratios computed on the training set. We can compute an optimal ROC curve for the combination by counting the number of true positives and false positives each rule achieves. In the test set we use the rules learned on the training set.

## 5.1 Results

The ROC graphs for our four experiments appear in Figure 2. The ROC curves in these experiments all rise very quickly and then flatten out, so we show only the range of $P_{F1}$ for which the values are interesting. We can draw some general conclusions from these graphs. First, PART clearly outperforms the other base classifiers in three out of four experiments, though it seems to overfit on the hypothyroid dataset. The LR-ROC dominates the ROC curves of the base classifiers on both training and test sets. The ROC curves for the base classifiers are all strictly below the LR-ROC in results on the test sets. The results on training sets seem to imply that the LR-ROC is primarily classifying like PART with a small boost from the other classifiers; however, the test set results (in particular, Figure 2b) demonstrate that the LR-ROC generalizes better than the base classifiers.

The robustness of our method to estimation errors is uncertain. In our experiments we found that smoothing did not improve generalization, but undoubtedly our method would benefit from better estimation of the outcome distribution and increased robustness.

We ran separate experiments to test how many classifiers our method could support in practice. Estimation of the joint distribution and computation of the ROC curve finished within one minute for 20 classifiers (not including time to train the individual classifiers). Unfortunately, the inherent exponential structure of the optimal ROC curve means we cannot expect to do significantly better (at the same rate, 30 classifiers would take over 12 hours and 40 classifiers almost a year and a half).

## 6 Related work

Our work is loosely related to ensemble methods such as *bagging* [8] and *boosting* [9] because it finds meta-classification rules over a set of base classifiers. However, bagging and boosting each take one base classifier and train many times, resampling or reweighting the training data to generate classifier diversity [10] or increase the classification margin [11]. The decision rules applied to the generated classifiers are (weighted) majority voting. In contrast, our method takes any binary classifiers and finds optimal combination rules from the more general space of all binary functions.

Ranking algorithms, such as RankBoost [12], approach the problem of ranking points by score or preference. Although we present our methods in a different light, our decision rule can be interpreted as a ranking algorithm. RankBoost, however, is a boosting algorithm and therefore fundamentally different from our approach. Ranking can be used for classification by choosing a cutoff or threshold, and in fact ranking algorithms tend to optimize the common Area Under the ROC Curve (AUC) metric. Although our method may have the side effect of maximizing the AUC, its formulation is different in that instead of optimizing a single global metric, it is a constrained optimization problem, maximizing $P_D$ for each $P_F$.

Another more similar method for combining classifiers is *stacking* [13]. Stacking trains a *meta-learner* to combine the predictions of several base classifiers; in fact, our method might be considered a stacking method with a particular meta-classifier. It can be difficult to show the improvement of stacking in general over selecting the best base-level classifier [14]; however, stacking has a useful interpretation as generalized cross-validation that makes it practical. Our analysis shows that our combination method is the optimal meta-learner in the Neyman-Pearson sense, but incorporating the model validation aspect of stacking would make an interesting extension to our work.

## 7 Conclusion

In this paper we introduce a new way to analyze a combination of classifiers and their ROC curves. We give a method for combining classifiers and prove that it is optimal in the Neyman-Pearson sense. This work raises several interesting questions.

Although the algorithm presented in this paper avoids checking the whole doubly exponential number of rules, the exponential factor in running time limits the number of classifiers that can be

combined in practice. Can a good approximation algorithm approach optimality while having lower time complexity? Though in general we make no assumptions about independence, Theorem 2 shows that certain simple rules are optimal when we do know that the classifiers are independent. Theorem 3 proves that the ROCCH can be optimal when only $n$ output combinations are possible. Perhaps other restrictions on the distribution of outcomes can lead to useful special cases.

## Acknowledgments

This work was supported in part by TRUST (Team for Research in Ubiquitous Secure Technology), which receives support from the National Science Foundation (NSF award number CCF-0424422) and the following organizations: AFOSR (#FA9550-06-1-0244), Cisco, British Telecom, ESCHER, HP, IBM, iCAST, Intel, Microsoft, ORNL, Pirelli, Qualcomm, Sun, Symantec, Telecom Italia, and United Technologies; and in part by the UC Berkeley-Taiwan International Collaboration in Advanced Security Technologies (iCAST) program. The opinions expressed in this paper are solely those of the authors and do not necessarily reflect the opinions of any funding agency or the U.S. or Taiwanese governments.

## References

[1] Foster Provost and Tom Fawcett. Robust classification for imprecise environments. *Machine Learning Journal*, 42(3):203–231, March 2001.

[2] Peter A. Flach and Shaomin Wu. Repairing concavities in ROC curves. In *Proceedings of the 19th International Joint Conference on Artificial Intelligence (IJCAI'05)*, pages 702–707, August 2005.

[3] Tom Fawcett. ROC graphs: Notes and practical considerations for data mining researchers. Technical Report HPL-2003-4, HP Laboratories, Palo Alto, CA, January 2003. Updated March 2004.

[4] J. Neyman and E. S. Pearson. On the problem of the most efficient tests of statistical hypotheses. *Philosophical Transactions of the Royal Society of London, Series A, Containing Papers of a Mathematical or Physical Character*, 231:289–337, 1933.

[5] Vincent H. Poor. *An Introduction to Signal Detection and Estimation*. Springer-Verlag, second edition, 1988.

[6] D. J. Newman, S. Hettich, C. L. Blake, and C. J. Merz. UCI repository of machine learning databases, 1998. http://www.ics.uci.edu/∼mlearn/MLRepository.html.

[7] I. Mierswa, M. Wurst, R. Klinkenberg, M. Scholz, and T. Euler. YALE: Rapid prototyping for complex data mining tasks. In *Proceedings of the ACM SIGKDD International Conference on Knowledge Discovery and Data Mining (KDD)*, 2006.

[8] L. Breiman. Bagging predictors. *Machine Learning*, 24(2):123–140, 1996.

[9] Y. Freund and R. E. Schapire. Experiments with a new boosting algorithm. In *Thirteenth International Conference on Machine Learning*, pages 148–156, Bari, Italy, 1996. Morgan Kaufmann.

[10] Thomas G. Dietterich. Ensemble methods in machine learning. *Lecture Notes in Computer Science*, 1857:1–15, 2000.

[11] Robert E. Schapire, Yoav Freund, Peter Bartlett, and Wee Sun Lee. Boosting the margin: A new explanation for the effectiveness of voting methods. *The Annals of Statistics*, 26(5):1651–1686, October 1998.

[12] Yoav Freund, Raj Iyer, Robert E. Schapire, and Yoram Singer. An efficient boosting algorithm for combining preferences. *Journal of Machine Learning Research (JMLR)*, 4:933–969, 2003.

[13] D. H. Wolpert. Stacked generalization. *Neural Networks*, 5:241–259, 1992.

[14] Saso Džeroski and Bernard Ženko. Is combining classifiers with stacking better than selecting the best one? *Machine Learning*, 54:255–273, 2004.

